# Entrainment of Silicon Central Pattern Generators for Legged Locomotory Control

**Francesco Tenore[1], Ralph Etienne-Cummings[1,2], M. Anthony Lewis[3]**
[1]Dept. of Electrical & Computer Eng., Johns Hopkins University, Baltimore, MD 21218
[2]Institute of Systems Research, University of Maryland, College Park, MD 20742
[3]Iguana Robotics, Inc., P.O. Box 625, Urbana, IL 61803

*{fra, retienne}@jhu.edu, tlewis@iguana-robotics.com*

## Abstract

We have constructed a second generation CPG chip capable of generating the necessary timing to control the leg of a walking machine. We demonstrate improvements over a previous chip by moving toward a significantly more versatile device. This includes a larger number of silicon neurons, more sophisticated neurons including voltage dependent charging and relative and absolute refractory periods, and enhanced programmability of neural networks. This chip builds on the basic results achieved on a previous chip and expands its versatility to get closer to a self-contained locomotion controller for walking robots.

## 1    Introduction

Legged locomotion is a system level behavior that engages most senses and activates most muscles in the human body. Understanding of biological systems is exceedingly difficult and usually defies any unifying analysis. Walking behavior is no exception. Theories of walking are likely incomplete, often in ways that are invisible to the scientist studying these behavior in animal or human systems. Biological systems often fill in gaps and details. One way of exposing our incomplete understanding is through the process of synthesis. In this paper we report on continued progress in building the basic elements of a motor pattern generator sufficient to control a legged robot. The focus of this paper is on a 2$^{nd}$ generation chip, that incorporates new features which we feel important for legged locomotion.

An essential element of most locomotory systems is the Central Patter Generator (CPG). The CPG is a set of neural circuits found in the spinal cord, arranged to produce oscillatory periodic waveforms that activate muscles in a coordinated manner. They are neuron primitives that are used in most periodic biological systems such as the respiratory, the digestive and the locomotory systems. In this last one, CPGs are constructed using neurons coupled together to produce phasic relationships required to achieve coordinated gait-type movements.

The CPG is more than a clock, or even a network of oscillators. Phenomena such as reflex reversal [7] can only be understood in terms of a system that has at least one additional state variable over sensory information alone. The CPG or similar circuits is certainly involved in modulation of sensory information from the periphery [5] and is of primary importance in providing phase information to the cerebellum. This information is necessary for coordination of the brain and the spinal cord [6].

Currently, there are two extremes in using CPGs for control of mechanical devices. The first is to be as faithful to the biological as possible, and then to discover how biological systems can assist in the control of complex machines. This approach is similar to that of Rasche et al. [1], based on the Hodgkin-Huxley model [3], and the one

implemented by Simoni and DeWeerth [2], based on the Morris-Lecar model [4]. These ion-channel based models imply a very large parameter space, making it difficult to work with in silicon, yet inviting direct comparison with biological counterparts.

Our approach is to start in the other direction. A system of minimal complexity was built [8,9] and then the question was asked of what additional features should be added to this minimal system to enable a behavior that is missing in the previous design. Thus, the two approaches start from different philosophical grounds, but will, hopefully, converge on similar solutions.

The motivation behind choosing a self-contained silicon system rather than a software implementation is that the former will use less power and be more compact and more amenable to the control of a power-autonomous robot.

Previously, a minimal system chip was built using integrate-and-fire neurons controlling a rudimentary robot [8, 9]. The chip described in this paper is an evolution of that one. Its main differences with its previous version are the following. The previous chip contained 2 spiking motoneurons and 2 pacemaker neurons, whereas the current chip contains 10 neurons of either type. More importantly, all the synapse weights (22 per neuron) are on-chip and can be used to make the synapse excitatory or inhibitory, while the previous version weighted the synapse signals outside the chip. The current chip also has 10 feedback synapses, making all the neurons interconnected. Moreover, the current chip has the capability of receiving and weighting up to 8 external inputs (instead of 2), such as sensory feedback signals, to allow better control of the CPG. The possibility of better tuning the pacemaker and spiking motoneurons created by the chip is achieved through direct modulation of the pulse width, of the absolute or relative refractory period and of the discharge strength of each neuron. Finally, the charging and discharging of the neurons' membrane capacitance is an exponential function of time, as opposed to the linear function that the previous chip exhibited. This allows for better coupling between CPGs (unpublished observation).

In this paper, after explaining the architecture of the chip and how simple networks can be created, a robotic application will be described. The paper will show that entrainment of multiple CPGs can be achieved by using direct coupling. Analysis and experiments demonstrating entrainment between multiple CPGs using direct coupling are presented. Finally, the oscillatory patterns used to control a single-legged robot are implemented in this chip.

## 2        Architecture

The CPG emulator chip was fabricated in silicon using a 0.5 μm CMOS process. The chip was designed to provide plausible electronic counterparts of biological elements, such as neurons, synapses, cell membranes, axons, and axon hillocks, for controlling motor systems. The chip also contains digital memories that can be used with synapses to modify weights or to modulate the membrane conductance. Through these components, it is possible to construct non-linear oscillators, which are based on the central pattern generators of biological organisms.

The chip's architecture can be seen in figure 1. It is made up of 10 fully interconnected "neurons" and 22 "synapses" per neuron. Communication with a particular neuron/synapse pair occurs through the address register, made up of the neuron/row register and the synapse/column register. Finally, a weight/data register allows a tunable amount of current to flow onto or away from the "neurons' axons."

Figure 2 shows a detailed view of a single neuron. As can be seen, all neurons are integrate-and-fire type neurons, in which the current that flows on the axon charges

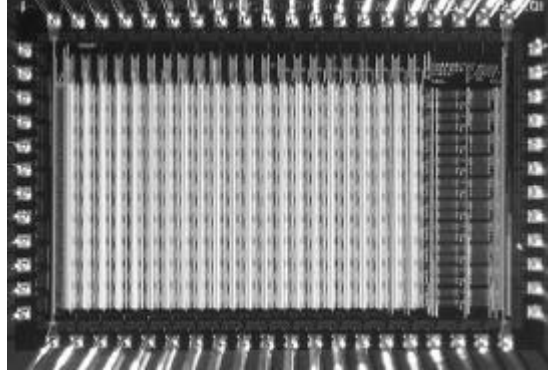

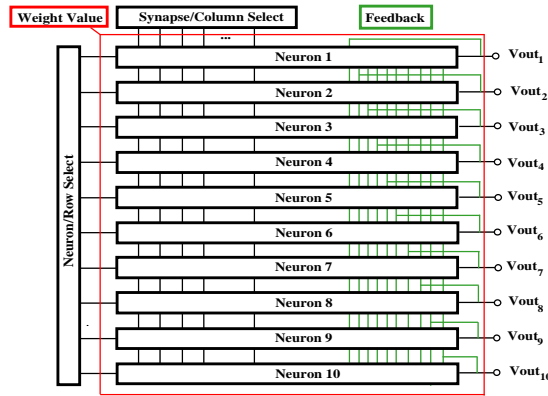

Figure 1. Top. Chip micrograph, 3.3x2.1 mm$^2$. The 22 synapses per neuron (vertical lines) are distinguishable. Bottom. System block diagram.

up the membrane capacitor, $C_{mem}$. When the voltage across the capacitor reaches a certain threshold, $V_{thresh}$, the hysteretic comparator output goes high. The output of the comparator does not change if the discharge and refractory period controls are disabled. Normally, however, the discharge controller is active and its function is to decrease the voltage on the membrane capacitance until it drops below the hysteretic comparator's lower threshold. The comparator output then goes low, the discharge is halted, and the capacitor can charge up again, thereby making the process start anew.

The $i$-th neuron can be modeled through the following set of equations:

$$C_i^{mem} \frac{dV_i^{mem}}{dt} = \sum_j W^+_{ij} I_j - \sum_k W^-_{ik} I_k - S_i I_{dis} - S_i I_{refrac} \qquad (1)$$

$$S_i(t+dt) = \begin{cases} 1 & if \quad (S_i(t)=1 \wedge V_i^{mem} > V_T^-) \vee (V_i^{mem} > V_T^+) \\ 0 & if \quad (S_i(t)=0 \wedge V_i^{mem} > V_T^+) \vee (V_i^{mem} > V_T^-) \end{cases} \qquad (2)$$

where $C_i^{mem}$ is the membrane capacitance of the $i$-th neuron, $V_T^+$ and $V_T^-$ are respectively the high and low thresholds of the hysteretic comparator, $V_i^{mem}$ is the voltage on the capacitor, $S_i(t)$ is the state of the hysteretic comparator at time $t$, $W^+_{ij}$ is the excitatory weight on the $j$-th excitatory synapse of the $i$-th neuron and similarly $W^-_{ik}$ is the inhibitory weight on the $k$-th inhibitory synapse of the $i$-th neuron. The discharge and refractory currents, $I_{dis}$ and $I_{refrac}$ correspond to the discharge and refractory period rates, respectively.

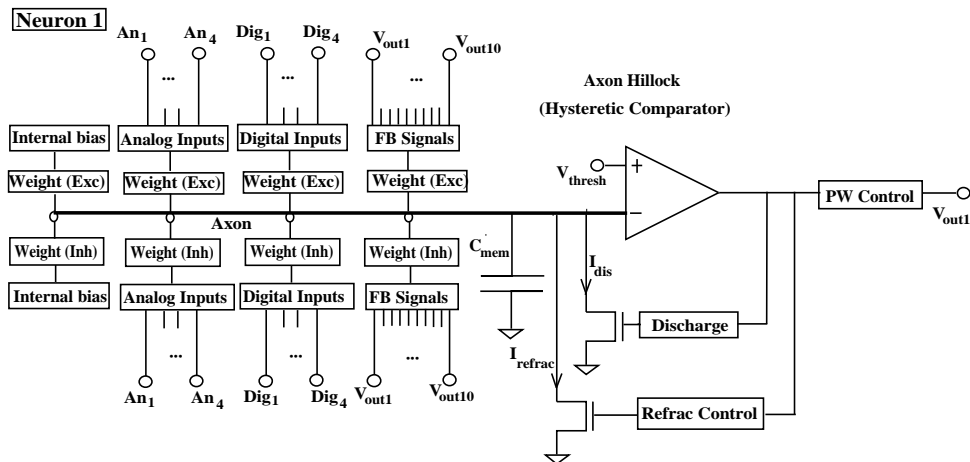

Figure 2. Block diagram of a single neuron. The neuron output is fed back to all the neurons including itself ($V_{out1}$ is also a feedback signal).

The speed with which the comparator changes state depends on the amount of current that the weight, or weights, sets on or remove from the "axon". The weights are set through 8-bit digital-to-analog converters (DACs) and stored in static random access memory (SRAM) cells. A ninth bit selects the type of weight, either excitatory or inhibitory. Finally, the three blocks that depend on the comparator output, work as follows. A weight can be set on any one of these three blocks, just as was done for the synapses. This allows modulation of the discharge strength, of the refractory period, and of the pulse width. The refractory period control element prevents current from charging up the capacitor for as long as it is active. It can be both relative and absolute, depending on its weight. The pulse-width block allows independent control of the output duty cycle by modifying the amount of time the output is high. As can be seen in figure 2, the output from the PW control block is both the neuron output and the feedback signal to all the neurons, including itself (self-feedback). The chip is thus fully interconnected.

From figure 2, four types of synapses can be identified. The first is the *internal bias* synapse, which allows current to flow onto or away from the membrane capacitor, depending on the type of bias it has, without requiring signals from inside the chip. The *analog* and *digital* synapses require the presence of an external analog or digital voltage to allow current to flow on the capacitor. The *feedback* synapses are also internal to the chip and allow the neurons to influence each other by modulating the charge-up of the membrane capacitors they are acting upon. This means that one of these synapses is of self-feedback for a particular neuron. These synapses are considered to be dual mode, in that they can both excite or inhibit. The 3 final synapses are used to control the discharge strength, the refractory period, and the pulse width.

It is thus possible to attain two types of waveforms at each neuron output, depending on the current charging the capacitor. If the current charges up and discharges the capacitor very quickly, the output is similar to that of a motor neuron. If the current charges and discharges the capacitor slowly, then the output is that of a pacemaker envelope neuron, which makes up the CPG.

## 3      Networks

Two simple networks are described in this section using this chip to understand the how the chip operates. The first example is shown in figure 3. A pacemaker neuron

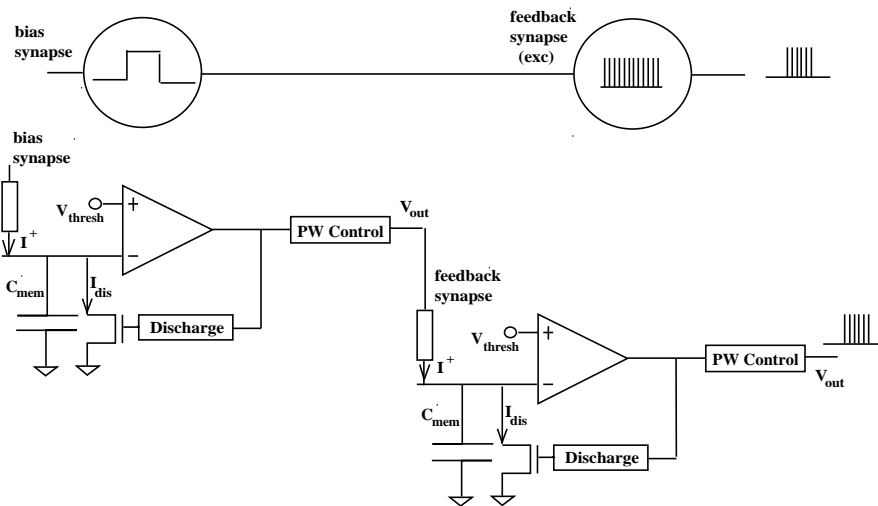

Figure 3. An envelope neuron exciting a motor neuron. The output waveforms are 180º out-of-phase.

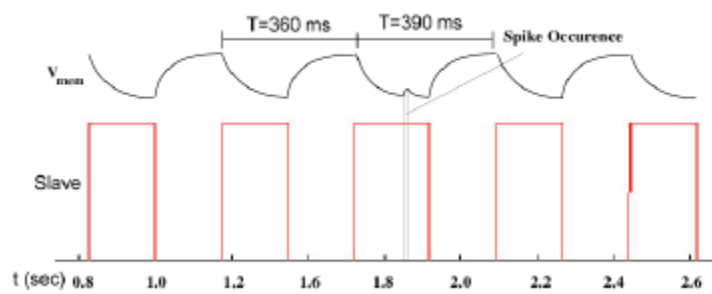

Figure 4. Master slave relationship. When the master spikes, the membrane potential increases for the duration of the spike.

controls the spiking of a motor neuron such that the spiking only occurs if the envelope is high. This is done using the internal biasing synapse to charge up the membrane capacitance of the envelope neuron and the feedback synapse coming from the envelope neuron to charge up the capacitor of the motor neuron. Similarly, the envelope neuron can inhibit the spiking which would otherwise occur at a constant rate through the bias synapse. Note that the bias synapse can either be the internally generated, as the one shown in figure 3, or it can be the one of the external analog or digital synapse seen in figure 2.

A second example, shown in figure 4, depicts the effects of a single spike on an envelope neuron. Depending on where the spike occurs with respect to the slave envelope neuron, it will either accelerate the charge-up or decelerate the discharge. In this example, the spike occurred during the membrane potential's discharge phase. The membrane potential's output voltage is shown within the slave output waveform. The two horizontal lines that delimit it represent the hysteretic comparator's threshold voltages. Thus, the slave stays high for a longer period of time, thus decreasing its normal frequency of oscillation. It is therefore possible to entrain the slave oscillator to the frequency of the master. This can be done either by increasing the duration of the master spike, increasing

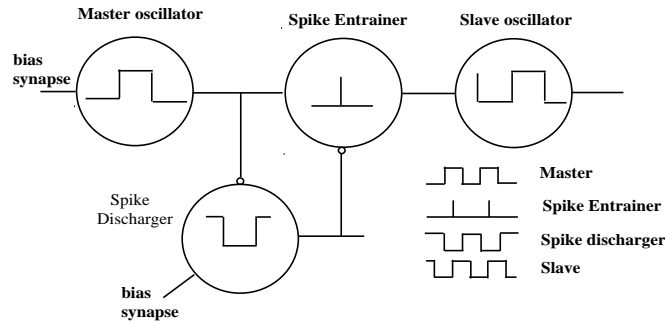

Figure 5. CPG entrainment.

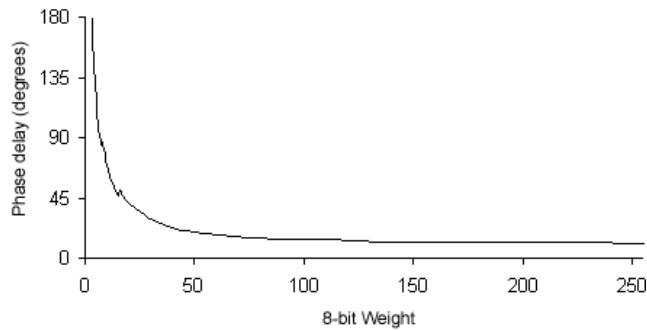

Figure 6. Phase delay between master envelope and spike entrainer.

the feedback weight with which the master controls the slave, or simply by increasing the spike frequency. For example, in this latter case, if the master frequency is higher than the slave's, then the spike will accelerate the slave such that it reaches the same period.

## 4        Analysis of pulse coupling

To show that it is possible to entrain two oscillators to have the same frequency but alter the phase at will, such that any phase between the two waveforms can be achieved, it is necessary to use a configuration similar to the one described in the previous section. A master and slave oscillator with different frequencies and both with approximately 50% duty cycle are set up as shown in figure 5. Another neuron is used to generate a single spike during the master's pulse width called the entrainer spike. It is evoked by the input from the master and has the same frequency, but its phase depends on the strength of the feedback synapse between these two cells. The spike's discharge occurs very slowly, but to ensure that no residual charge is left on the capacitor, a fourth neuron, 180º out-of-phase with the master, is used. When this neuron is high, it sends a strong inhibition signal to the spike, thereby resetting it. At this point, the spike can be used for synchronizing the slave oscillator. As described previously, if the slave oscillator's frequency is lower than the master's (and therefore that of the spike's), the spike's effect is to accelerate until the two are synchronized. This allows for two pacemaker neurons to be out-of-phase by an arbitrary angle. This is shown in figure 6, where the coupling weight between master and slave was systematically altered and the resulting phase variation was recorded. To fine tune the slave oscillator's desired phase difference, once the spike master has been set, it is necessary to tune the feedback strength between the spike and the slave oscillator. A stronger feedback will allow the

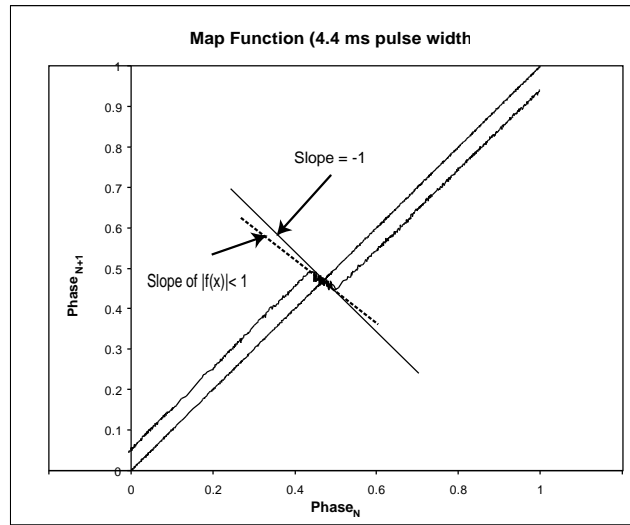

Figure 7. Map function illustrating the coupling behavior between two neurons.

two signals to happen virtually at the same time, a weaker weight will cause some delay between the two. Lewis and Bekey show that adaptation of time is critical to controlling walking in a robot [10].

Finally, figure 7 shows a map function obtained using a 4.4 ms spike pulse width. A map function depicts the effect of a spike on a pacemaker neuron at all possible phases. The curve shows a slope smaller 1 (in absolute value) in the transition region, which implies that the system is asymptotically stable [9].

## 5       Experiment

To build on all the results achieved, the oscillatory patterns necessary to control a single-legged robot were synthesized. Figure 7 shows the waveforms generated to control a hip's flexor and extensor muscles and ipsilateral knee's flexor and extensor. These waveforms were generated using all 10 available neurons with the procedures described previously. The hip flexor and extensor are 180º out-of-phase to each other. The left knee extensor is slightly out-of-phase with its respective hip muscle but the width of the waveform's pulse is shorter than that of the hip extensor. As can be seen, the knee flexor has two bumps, where the purpose of the first bump is to stabilize the knee when the foot hits the substrate. The waveforms depicted are necessary to drive a robotic leg with a standard walking gait. Different gaits will have waveforms with different phase relationships. However, the results shown in the previous sections show that these waveforms, through simple variations of the timing parameters described, can be generated with ease.

## 6       Conclusions

The waveforms needed to control a robotic leg can be generated using a silicon chip described in this paper. The phase differences between the waveforms, however, change depending on the type of gait that one wants to implement in a robot. The results obtained show that any phase difference between two or more waveforms can be achieved, thus making any gait effectively achievable. Furthermore, the map function that

resulted from on-chip measurements showed that the chip has the capability of asymptotic coupling stability.

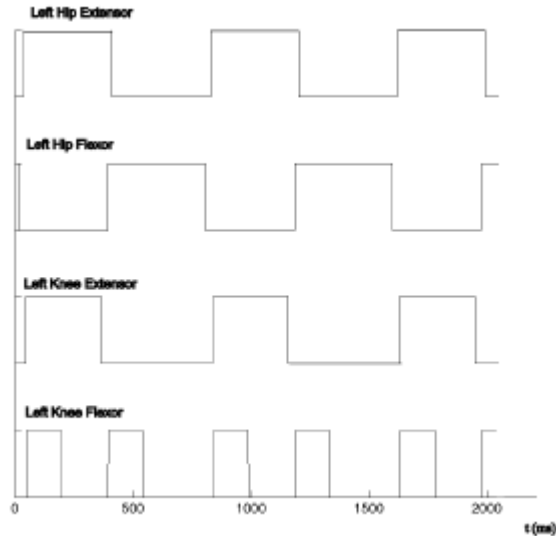

Figure 8. Waveforms generated to control a robotic leg.

**References**
[1]. C. Rasche, R. Douglas, M. Mahowald, "Characterization of a pyramidal silicon neuron," *Neuromorphic Systems: Engineering silicon from neurobiology*, L. S. Smith and A. Hamilton, eds, World Scientific, 1st edition, 1998.
[2]. M. Simoni, S. DeWeerth, "Adaptation in an aVLSI model of a neuron." *IEEE Transactions on circuits and systems II: Analog and digital signal processing*. 46(7):967-970, 1999.
 [3]. A.L. Hodgkin, A.F. Huxley. "A quantitative description of ion currents and its applications to conduction and excitation in nerve membranes," *Journal of Physiology (Lond.)*, 117:500-544, 1952.
[4]. C. Morris, H. Lecar, "Voltage oscillations in the barnacle giant muscle fiber," *Biophysics J.*, vol. 35, pp. 193-213, 1981.
[5]. Y.I. Arshavsky, I. M. Gelfand, and G. N. Orlovsky, "The cerebellum and control of rhythmic movements," *TINS*, vol. 6, pp. 417-422, 1983.
[6]. A.H. Cohen, D.L. Boothe, "Sensorimotor interactions during locomotion: principles derived from biological systems," *Autonomous robots, special issue on biomorphic robots*, M.A. Lewis and M.A. Arbib, (Eds). Vol. 7, pp. 225-238, 1999.
[7]. H. Forssberg, S. Grillner, S. Rossignol, "Phase dependent reflex during walking in chronic spinal cats," *Brain research*, vol. 85, pp. 103-7, 1975.
 [8]. M.A. Lewis, R. Etienne-Cummings, A.H. Cohen, M. Hartmann, "Toward biomorphic control using custom aVLSI chips", *Proceedings of the International conference on robotics and automation*, San Francisco, CA, 2000.
[9]. M. A. Lewis, R. Etienne-Cummings, M. J. Hartmann, A. H. Cohen, Z. R. Xu, "An *in silico* central pattern generator: silicon oscillator, coupling, entrainment, and physical computation", *Biological Cybernetics*, 88, 2, 2003, pp. 137-151.
[10]. M. Anthony Lewis and George A. Bekey (2002), Gait Adaptation in a Quadruped robot, *Autonomous Robots*, 12(3) 301-312.
